# Noisy Spiking Neurons with Temporal Coding have more Computational Power than Sigmoidal Neurons

**Wolfgang Maass**
Institute for Theoretical Computer Science
Technische Universitaet Graz, Klosterwiesgasse 32/2
A-8010 Graz, Austria, e-mail: maass@igi.tu-graz.ac.at

## Abstract

We exhibit a novel way of simulating sigmoidal neural nets by networks of noisy spiking neurons in temporal coding. Furthermore it is shown that networks of noisy spiking neurons with temporal coding have a strictly larger computational power than sigmoidal neural nets with the same number of units.

## 1 Introduction and Definitions

We consider a formal model SNN for a spiking neuron network that is basically a reformulation of the spike response model (and of the leaky integrate and fire model) without using $\delta$-functions (see [Maass, 1996a] or [Maass, 1996b] for further background).

An SNN consists of a finite set $V$ of *spiking neurons*, a set $E \subseteq V \times V$ of *synapses*, a *weight* $w_{u,v} \geq 0$ and a *response function* $\varepsilon_{u,v} : \mathbf{R}^+ \to \mathbf{R}$ for each synapse $\langle u, v \rangle \in E$ (where $\mathbf{R}^+ := \{x \in \mathbf{R} : x \geq 0\}$), and a *threshold function* $\Theta_v : \mathbf{R}^+ \to \mathbf{R}^+$ for each neuron $v \in V$.

If $F_u \subseteq \mathbf{R}^+$ is the set of *firing times* of a neuron $u$, then the *potential* at the trigger zone of neuron $v$ at time $t$ is given by

$$P_v(t) := \sum_{u:\langle u,v \rangle \in E} \sum_{s \in F_u : s < t} w_{u,v} \cdot \varepsilon_{u,v}(t - s) .$$

In a noise-free model a neuron $v$ fires at time $t$ as soon as $P_v(t)$ reaches $\Theta_v(t - t')$, where $t'$ is the time of the most recent firing of $v$. One says then that neuron $v$ sends out an "action potential" or "spike" at time $t$.

For some specified subset $V_{in} \subseteq V$ of *input neurons* one assumes that the firing times ("spike trains") $F_u$ for neurons $u \in V_{in}$ are not defined by the preceding convention, but are given from the outside. The firing times $F_v$ for all other neurons $v \in V$ are determined by the previously described rule, and the output of the network is given in the form of the spike trains $F_v$ for a specified set of *output neurons* $V_{out} \subseteq V$ .

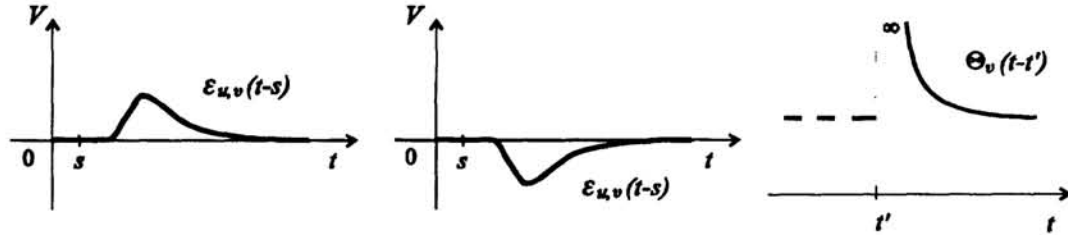

*Figure 1: Typical shapes of response functions $\varepsilon_{u,v}$ (EPSP and IPSP) and threshold functions $\Theta_v$ for biological neurons.*

We will assume in our subsequent constructions that all response functions $\varepsilon_{u,v}$ and threshold functions $\Theta_v$ in an SNN are "stereotyped", i.e. that the response functions differ apart from their "sign" (EPSP or IPSP) only in their delay $d_{u,v}$ (where $d_{u,v} := \inf \{t \geq 0 : \varepsilon_{u,v}(t) \neq 0\}$) , and that the threshold functions $\Theta_v$ only differ by an additive constant (i.e. for all $u$ and $v$ there exists a constant $c_{u,v}$ such that $\Theta_u(t) = \Theta_v(t) + c_{u,v}$ for all $t \geq 0$) . We refer to a term of the form $w_{u,v} \cdot \varepsilon_{u,v}(t - s)$ as an excitatory respectively inhibitory postsynaptic potential (abbreviated: EPSP respectively IPSP).

Since biological neurons do not always fire in a reliable manner one also considers the related model of *noisy spiking neurons*, where $P_v(t)$ is replaced by $P_v^{noisy}(t) := P_v(t) + \alpha_v(t)$ and $\Theta_v(t - t')$ is replaced by $\Theta_v^{noisy}(t - t') := \Theta_v(t - t') + \beta_v(t - t')$. $\alpha_v(t)$ and $\beta_v(t - t')$ are allowed to be arbitrary functions with bounded absolute value (hence they can also represent "systematic noise").

Furthermore one allows that the current value of the difference $D(t) := P_v^{noisy}(t) - \Theta_v^{noisy}(t - t')$ does not determine directly the firing time of neuron $v$, but only its current *firing probability*. We assume that the firing probability approaches 1 if $D \to \infty$ , and 0 if $D \to -\infty$ . We refer to spiking neurons with these two types of noise as "noisy spiking neurons".

We will explore in this article the power of *analog* computations with noisy spiking neurons, and we refer to [Maass, 1996a] for results about *digital* computations in this model. Details to the results in this article appear in [Maass, 1996b] and [Maass, 1997].

## 2   Fast Simulation of Sigmoidal Neural Nets with Noisy Spiking Neurons in Temporal Coding

So far one has only considered simulations of sigmoidal neural nets by spiking neurons where each analog variable in the sigmoidal neural net is represented by the *firing rate* of a spiking neuron. However this "firing rate interpretation" is inconsistent with a number of empirical results about computations in biological neural

systems. For example [Thorpe & Imbert, 1989] have demonstrated that visual pattern analysis and pattern classification can be carried out by humans in just 150 ms, in spite of the fact that it involves a minimum of 10 synaptic stages from the retina to the temporal lobe. [de Ruyter van Stevenick & Bialek, 1988] have found that a blowfly can produce flight torques within 30 ms of a visual stimulus by a neural system with several synaptic stages. However the firing rates of neurons involved in all these computations are usually below 100 Hz, and interspike intervals tend to be quite irregular. Hence one cannot interpret these analog computations with spiking neurons on the basis of an encoding of analog variables by firing rates.

On the other hand experimental evidence has accumulated during the last few years which indicates that many biological neural systems use the *timing* of action potentials to encode information (see e.g. [Bialek & Rieke, 1992], [Bair & Koch, 1996]).

We will now describe a new way of simulating sigmoidal neural nets by networks of spiking neurons that is based on *temporal coding*. The *key mechanism* for this alternative simulation is based on the well known fact that EPSP's and IPSP's are able to *shift* the firing time of a spiking neuron. This mechanism can be demonstrated very clearly in our formal model if one assumes that EPSP's rise (and IPSP's fall) *linearly* during a certain initial time period. Hence we assume in the following that there exists some constant $\Delta > 0$ such that each response function $\varepsilon_{u,v}(x)$ is of the form $\alpha_{u,v} \cdot (x - d_{u,v})$ with $\alpha_{u,v} \in \{-1, 1\}$ for $x \in [d_{u,v}, d_{u,v} + \Delta]$, and $\varepsilon_{u,v}(x) = 0$ for $x \in [0, d_{u,v}]$.

Consider a spiking neuron $v$ that receives postsynaptic potentials from $n$ presynaptic neurons $a_1, \ldots, a_n$. For simplicity we assume that interspike intervals are so large that the firing time $t_v$ of neuron $v$ depends just on a single firing time $t_{a_i}$ of each neuron $a_i$, and $\Theta_v$ has returned to its "resting value" $\Theta_v(0)$ before $v$ fires again. Then if the next firing of $v$ occurs at a time when the postsynaptic potentials described by $w_{a_i,v} \cdot \varepsilon_{a_i,v}(t - t_{a_i})$ are all in their initial *linear* phase, its firing time $t_v$ is determined in the noise-free model for $w_i := w_{a_i,v} \cdot \alpha_{a_i,v}$ by the equation $\sum_{i=1}^{n} w_i \cdot (t_v - t_{a_i} - d_{a_i,v}) = \Theta_v(0)$, or equivalently

$$t_v = \frac{\Theta_v(0)}{\sum_{i=1}^{n} w_i} + \frac{\sum_{i=1}^{n} w_i \cdot (t_{a_i} + d_{a_i,v})}{\sum_{i=1}^{n} w_i} \tag{1}$$

This equation reveals the somewhat surprising fact that (for a certain range of their parameters) spiking neurons can compute a *weighted sum* in terms of *firing times*, i.e. *temporal coding*. One should also note that in the case where all delays $d_{a_i,v}$ have the same value, the "weights" $w_i$ of this weighted sum are encoded in the "strengths" $w_{a_i,v}$ of the synapses and their "sign" $\alpha_{a_i,v}$, as in the "firing rate interpretation". Finally according to (1) the coefficients of the presynaptic firing times $t_{a_i}$ are automatically *normalized*, which appears to be of biological interest.

In the simplest scheme for temporal coding (which is closely related to that in [Hopfield, 1995]) an analog variable $x \in [0, 1]$ is encoded by the firing time $T - \gamma \cdot x$ of a neuron, where $T$ is assumed to be independent of $x$ (in a biological context $T$ might be time-locked to the onset of a stimulus, or to some oscillation) and $\gamma$ is some constant that is determined in the proof of Theorem 2.1 (e.g. $\gamma = \Delta/2$ in the noise-free case). In contrast to [Hopfield, 1995] we assume that both the inputs *and the outputs* of computations are encoded in this fashion. This has the advantage that one can *compose* computational modules.

We will first focus in Theorem 2.1 on the simulation of sigmoidal neural nets that employ the piecewise linear "linear saturated" activation function $\pi : \mathbf{R} \to [0, 1]$ defined by $\pi(y) = 0$ if $y < 0$, $\pi(y) = y$ if $0 \le y \le 1$, and $\pi(y) = 1$ if $y > 1$. The Theorem 3.1 in the next section will imply that one can simulate with spiking neurons also sigmoidal neural nets that employ *arbitrary* continuous activation functions. Apart from the previously mentioned assumptions we will assume for the proofs of Theorem 2.1 and 3.1 that any EPSP satisfies $\varepsilon_{u,v}(x) = 0$ for all sufficiently large $x$, and $\varepsilon_{u,v}(x) \ge \varepsilon_{u,v}(d_{u,v} + \Delta)$ for all $x \in [d_{u,v} + \Delta, d_{u,v} + \Delta + \gamma]$. We assume that each IPSP is continuous, and has value 0 except for some interval of $\mathbf{R}$. Furthermore we assume for each EPSP and IPSP that $|\varepsilon_{u,v}(x)|$ grows at most linearly during the interval $[d_{u,v} + \Delta, d_{u,v} + \Delta + \gamma]$. In addition we assume that $\Theta_v(x) = \Theta_v(0)$ for sufficiently large $x$, and that $\Theta_v(x)$ is sufficiently large for $0 < x \le \gamma$.

**Theorem 2.1** *For any given $\varepsilon, \delta > 0$ one can simulate any given feedforward sigmoidal neural net $N$ with activation function $\pi$ by a network $\mathcal{N}_{N,\varepsilon,\delta}$ of noisy spiking neurons in temporal coding. More precisely, for any network input $x_1, \dots, x_m \in [0, 1]$ the output of $\mathcal{N}_{N,\varepsilon,\delta}$ differs with probability $\ge 1 - \delta$ by at most $\varepsilon$ from that of $N$. Furthermore the computation time of $\mathcal{N}_{N,\varepsilon,\delta}$ depends neither on the number of gates in $N$ nor on the parameters $\varepsilon, \delta$, but only on the number of* *layers* *of the sigmoidal neural network $N$.*

We refer to [Maass, 1997] for details of the somewhat complicated proof. One employs the mechanism described by (1) to simulate through the firing time of a spiking neuron $v$ a sigmoidal gate with activation function $\pi$ for those gate-inputs where $\pi$ operates in its linearly rising range. With the help of an auxiliary spiking neuron that fires at time $T$ one can avoid the automatic "normalization" of the weights $w_i$ that is provided by (1), and thereby compute a weighted sum with *arbitrary* given weights. In order to simulate in temporal coding the behaviour of the gate in the input range where $\pi$ is "saturated" (i.e. constant), it suffices to employ some auxiliary spiking neurons which make sure that $v$ fires exactly once during the relevant time window (and not shortly before that).

Since inputs and outputs of the resulting modules for each single gate of $N$ are all given in temporal coding, one can compose these modules to simulate the multi-layer sigmoidal neural net $N$. With a bit of additional work one can ensure that this construction also works with *noisy* spiking neurons.                                                                        ∎

## 3   Universal Approximation Property of Networks of Noisy Spiking Neurons with Temporal Coding

It is known [Leshno et al., 1993] that feedforward sigmoidal neural nets whose gates employ the activation function $\pi$ can approximate with a single hidden layer for any $n, k \in \mathbf{N}$ any given continuous function $F : [0, 1]^n \to [0, 1]^k$ within any $\varepsilon > 0$ with regard to the $L_\infty$-norm (i.e. uniform convergence). Hence we can derive the following result from Theorem 2.1:

**Theorem 3.1** *Any given continuous function $F : [0, 1]^n \to [0, 1]^k$ can be approximated within any given $\varepsilon > 0$ with arbitrarily high reliability in temporal coding by*

*a network of noisy spiking neurons (SNN) with a single hidden layer (and hence within 15 ms for biologically realistic values of their time-constants).* ∎

Because of its generality this Theorem implies the same result also for *more general schemes of coding analog variables by the firing times of neurons*, besides the particular one that we have considered so far. In fact it implies that the same result holds for any other coding scheme $C$ that is "continuously related" to the previously considered one in the sense that the transformation between firing times that encode an analog variable $x$ in the here considered coding scheme and in the coding scheme $C$ can be described by uniformly continuous functions in both directions.

## 4  Spiking Neurons have more Computational Power than Sigmoidal Neurons

We consider the *"element distinctness function"* $ED_n : (\mathbf{R}^+)^n \to \{0,1\}$ defined by

$$ED_n(s_1,\ldots,s_n) = \begin{cases} 1, & \text{if } s_i = s_j \text{ for some } i \neq j \\ 0, & \text{if } |s_i - s_j| \geq 1 \text{ for all } i,j \text{ with } i \neq j \\ \text{arbitrary,} & \text{else .} \end{cases}$$

If one encodes the value of input variable $s_i$ by a firing of input neuron $a_i$ at time $T_{in} - c \cdot s_i$, then for sufficiently large values of the constant $c > 0$ *a single noisy spiking neuron $v$ can compute $ED_n$ with arbitrarily high reliability*. This holds for any reasonable type of response functions, e.g. the ones shown in Fig. 1. The binary output of this computation is assumed to be encoded by the firing/non-firing of $v$. Hair-trigger situations are avoided since no assumptions have to be made about the firing or non-firing of $v$ if EPSP's arrive with a temporal distance *between* 0 and $c$.

On the other hand the following result shows that a fairly large *sigmoidal* neural net is needed to compute the same function. Its proof provides the first application for Sontag's recent results about a new type of "dimension" $d$ of a neural network $N$, where $d$ is chosen maximal so that *every* subset of $d$ inputs is shattered by $N$. Furthermore it expands a method due to [Koiran, 1995] for using the VC-dimension to prove lower bounds on network size.

**Theorem 4.1** *Any sigmoidal neural net $N$ that computes $ED_n$ has at least $\frac{n-4}{2} - 1$ hidden units.*

**Proof:** Let $N$ be an arbitrary sigmoidal neural net with $k$ gates that computes $ED_n$. Consider *any* set $S \subseteq \mathbf{R}^+$ of size $n - 1$. Let $\lambda > 0$ be sufficiently large so that the numbers in $\lambda \cdot S$ have pairwise distance $\geq 2$. Let $A$ be a set of $n - 1$ numbers $> \max(\lambda \cdot S) + 2$ with pairwise distance $\geq 2$.

By assumption $N$ can decide for $n$ arbitrary inputs from $\lambda \cdot S \cup A$ whether they are all different. Let $N_\lambda$ be a variation of $N$ where all weights on edges from the first input variable are multiplied with $\lambda$. Then $N_\lambda$ can compute any function from

$S$ into $\{0, 1\}$ after one has assigned a suitable fixed set of $n-1$ pairwise different numbers from $\lambda \cdot S \cup A$ to the last $n-1$ input variables.

Thus if one considers as *programmable* parameters of $\mathcal{N}$ the factor $\lambda$ in the weights on edges from the first input variable and the $\leq k$ thresholds of gates that are connected to some of the other $n-1$ input variables, then $\mathcal{N}$ shatters $S$ with these $k+1$ programmable parameters.

Since $S \subseteq \mathbf{R}^+$ of size $n-1$ was chosen *arbitrarily*, we can now apply the result from [Sontag, 1996], which yields an upper bound of $2w + 1$ for the maximal number $d$ such that *every* set of $d$ different inputs can be shattered by a sigmoidal neural net with $w$ programmable parameters (note that this parameter $d$ is in general much smaller than the VC-dimension of the neural net). For $w := k + 1$ this implies in our case that $n - 1 \leq 2(k+1) + 1$, hence $k \geq (n-4)/2$. Thus $\mathcal{N}$ has at least $(n-4)/2$ computation nodes, and therefore at least $(n-4)/2 - 1$ hidden units. One should point out that due to the generality of Sontag's result this lower bound is valid for all common activation functions of sigmoidal gates, and even if $\mathcal{N}$ employs heaviside gates besides sigmoidal gates.                                     ∎

Theorem 4.1 yields a lower bound of 4997 for the number of hidden units in any sigmoidal neural net that computes $ED_n$ for $n = 10\,000$, where $10\,000$ is a common estimate for the number of inputs (i.e. synapses) of a biological neuron.

Finally we would like to point out that to the best of our knowledge Theorem 4.1 provides the largest known lower bound for *any* concrete function with $n$ inputs on a sigmoidal neural net. The largest previously known lower bound for sigmoidal neural nets was $\Omega(n^{1/4})$, due to [Koiran, 1995].

## 5  Conclusions

Theorems 2.1 and 3.1 provide a model for analog computations in network of spiking neurons that is consistent with experimental results on the maximal computation speed of biological neural systems. As explained after Theorem 3.1, this result holds for a large variety of possible schemes for encoding analog variables by firing times.

These theoretical results hold *rigorously* only for a rather small time window of length $\gamma$ for temporal coding. However a closer inspection of the construction shows that the actual shape of EPSP's and IPSP's in biological neurons provides an automatic adjustment of extreme values of the inputs $t_{a_i}$ towards their average, which allows them to carry out rather similar computations for a substantially larger window size. It also appears to be of interest from the biological point of view that the synaptic weights play for temporal coding in our construction basically the same role as for rate coding, and hence the *same* network is in principle able to compute closely related analog functions in *both* coding schemes.

We have focused in our constructions on feedforward nets, but our method can for example also be used to simulate a Hopfield net with graded response by a network of noisy spiking neurons in temporal coding. A stable state of the Hopfield net corresponds then to a firing pattern of the simulating SNN where all neurons fire at the same frequency, with the *"pattern"* of the stable state encoded in their phase differences.

The theoretical results in this article may also provide additional goals and directions for a new computer technology based on *artificial* spiking neurons.

## Acknowledgement

I would like to thank David Haussler, Pascal Koiran, and Eduardo Sontag for helpful communications.

## References

[Bair & Koch, 1996] W. Bair, C. Koch, "Temporal precision of spike trains in extrastriate cortex of the behaving macaque monkey", *Neural Computation*, vol. 8, pp 1185–1202, 1996.

[Bialek & Rieke, 1992] W. Bialek, and F. Rieke, "Reliability and information transmission in spiking neurons", *Trends in Neuroscience*, vol. 15, pp 428–434, 1992.

[Hopfield, 1995] J. J. Hopfield, "Pattern recognition computation using action potential timing for stimulus representations", *Nature*, vol. 376, pp 33–36, 1995.

[Koiran, 1995] P. Koiran, "VC-dimension in circuit complexity", *Proc. of the 11th IEEE Conference on Computational Complexity*, pp 81–85, 1996.

[Leshno et al., 1993] M. Leshno, V. Y. Lin, A. Pinkus, and S. Schocken, "Multilayer feedforward networks with a nonpolynomial activation function can approximate any function", *Neural Networks*, vol. 6, pp 861–867, 1993.

[Maass, 1996a] W. Maass, "On the computational power of noisy spiking neurons", *Advances in Neural Information Processing Systems*, vol. 8, pp 211-217, MIT Press, Cambridge, 1996.

[Maass, 1996b] W. Maass, "Networks of spiking neurons: the third generation of neural network models", FTP-host: archive.cis.ohio-state.edu, FTP-filename: /pub/neuroprose/maass.third-generation.ps.Z, *Neural Networks*, to appear.

[Maass, 1997] W. Maass, "Fast sigmoidal networks via spiking neurons", to appear in Neural Computation. FTP-host: archive.cis.ohio-state.edu FTP-filename: /pub/neuroprose/maass.sigmoidal-spiking.ps.Z, *Neural Computation*, to appear in vol. 9, 1997.

[de Ruyter van Steveninck & Bialek, 1988] R. de Ruyter van Steveninck, and W. Bialek, "Real-time performance of a movement sensitive neuron in the blowfly visual system", *Proc. Roy. Soc. B*, vol. 234, pp 379–414, 1988.

[Sontag, 1996] E. D. Sontag, "Shattering all sets of $k$ points in 'general position' requires $(k-1)/2$ parameters", http://www.math.rutgers.edu/~sontag/ , follow links to FTP archive.

[Thorpe & Imbert, 1989] S. T. Thorpe, and M. Imbert, "Biological constraints on connectionist modelling", In: *Connectionism in Perspective*, R. Pfeifer, Z. Schreter, F. Fogelman-Soulié, and L. Steels, eds., Elsevier, North-Holland, 1989.